# A primal-dual algorithm for group sparse regularization with overlapping groups

**Sofia Mosci**
DISI- Università di Genova
mosci@disi.unige.it

**Silvia Villa**
DISI- Università di Genova
villa@dima.unige.it

**Alessandro Verri**
DISI- Università di Genova
verri@disi.unige.it

**Lorenzo Rosasco**
IIT - MIT
lrosasco@MIT.EDU

## Abstract

We deal with the problem of variable selection when variables must be selected group-wise, with possibly overlapping groups defined a priori. In particular we propose a new optimization procedure for solving the regularized algorithm presented in [12], where the group lasso penalty is generalized to overlapping groups of variables. While in [12] the proposed implementation requires explicit replication of the variables belonging to more than one group, our iterative procedure is based on a combination of proximal methods in the primal space and projected Newton method in a reduced dual space, corresponding to the active groups. This procedure provides a scalable alternative with no need for data duplication, and allows to deal with high dimensional problems without pre-processing for dimensionality reduction. The computational advantages of our scheme with respect to state-of-the-art algorithms using data duplication are shown empirically with numerical simulations.

## 1   Introduction

Sparsity has become a popular way to deal with small samples of high dimensional data and, in a broad sense, refers to the possibility of writing the solution in terms of a few building blocks. Often, sparsity based methods are the key towards finding interpretable models in real-world problems. In particular, regularization based on $\ell_1$ type penalties is a powerful approach for dealing with the problem of variable selection, since it provides sparse solutions by minimizing a convex functional. The success of $\ell_1$ regularization motivated exploring different kinds of sparsity properties for (generalized) linear models, exploiting available a priori information, which restricts the admissible sparsity patterns of the solution. An example of a sparsity pattern is when the input variables are partitioned into groups (known a priori), and the goal is to estimate a sparse model where variables belonging to the same group are either jointly selected or discarded. This problem can be solved by regularizing with the group-$\ell_1$ penalty, also known as group lasso penalty, which is the sum, over the groups, of the euclidean norms of the coefficients restricted to each group.

A possible generalization of group lasso is to consider groups of variables which can be potentially overlapping, and the goal is to estimate a model which support is the union of groups. This is a common situation in bioinformatics (especially in the context of high-throughput data such as gene expression and mass spectrometry data), where problems are characterized by a very low number of samples with several thousands of variables. In fact, when the number of samples is not sufficient to guarantee accurate model estimation, an alternative is to take advantage of the huge amount of prior knowledge encoded in online databases such as the Gene Ontology. Largely motivated by applications in bioinformatics, a new type of penalty is proposed in [12], which is shown to give better

performances than simple $\ell_1$ regularization.

A straightforward solution to the minimization problem underlying the method proposed in [12] is to apply state-of-the-art techniques for group lasso (we recall interior-points methods [3, 20], block coordinate descent [16], and proximal methods [9, 21], also known as forward-backward splitting algorithms, among others) in an expanded space, built by duplicating variables that belong to more than one group.

As already mentioned in [12], though very simple, such an implementation does not scale to large datasets, when the groups have significant overlap, and a more scalable algorithm with no data duplication is needed. For this reason we propose an alternative optimization approach to solve the group lasso problem with overlap. Our method does not require explicit replication of the features and is thus more appropriate to deal with high dimensional problems with large groups overlap. Our approach is based on a proximal method (see for example [18, 6, 5]), and two ad hoc results that allow to efficiently compute the proximity operator in a much lower dimensional space: with Lemma 1 we identify the subset of *active* groups, whereas in Theorem 2 we formulate the reduced dual problem for computing the proximity operator, where the dual space dimensionality coincides with the number of active groups. The dual problem can then be solved via Bertsekas' projected Newton method [7]. We recall that a particular overlapping structure is the hierarchical structure, where the overlap between groups is limited to inclusion of a descendant in its ancestors. In this case the CAP penalty [24] can be used for model selection, as it has been done in [2, 13], but ancestors are forced to be selected when any of their descendant are selected. Thanks to the nested structure, the proximity operator of the penalty term can be computed exactly in a finite number of steps [14]. This is no longer possible in the case of general overlap. Finally it is worth noting that the penalty analyzed here can be applied also to hierarchical group lasso. Differently from [2, 13] selection of ancestors is no longer enforced.

The paper is organized as follows. In Section 2 we recall the group lasso functional for overlapping groups and set some notations. In Section 3 we state the main results, present a new iterative optimization procedure, and discuss computational issues. Finally in Section 4 we present some numerical experiments comparing running time of our algorithm with state-of-the-art techniques. The proofs are reported in the Supplementary material.

## 2 Problem and Notations

We first fix some notations. Given a vector $\beta \in \mathbb{R}^d$, while $\|\cdot\|$ denotes the $\ell_2$-norm, we will use the notation $\|\beta\|_G = (\sum_{j \in G} \beta_j^2)^{1/2}$ to denote the $\ell_2$-norm of the components of $\beta$ in $G \subset \{1, \dots, d\}$. Then, for any differentiable function $f : \mathbb{R}^B \to \mathbb{R}$, we denote by $\partial_r f$ its partial derivative with respect to variables $r$, and by $\nabla f = (\partial_r f)_{r=1}^B$ its gradient.

We are now ready to cast group $\ell_1$ regularization with overlapping groups as the following variational problem. Given a training set $\{(x_i, y_i)_{i=1}^n\} \in (X \times Y)^n$, a dictionary $(\psi_j)_{j=1}^d$, and $B$ subsets of variables $\mathcal{G} = \{G_r\}_{r=1}^B$ with $G_r \subset \{1, \dots, d\}$, we assume the estimator to be described by a generalized linear model $f(x) = \sum_{j=1}^d \psi_j(x)\beta_j$ and consider the following regularization scheme

$$\beta^* = \operatorname*{argmin}_{\beta \in \mathbb{R}^d} \mathcal{E}_\tau(\beta) = \operatorname*{argmin}_{\beta \in \mathbb{R}^d} \left\{ \frac{1}{n} \|\Psi\beta - y\|^2 + 2\tau \Omega_{\text{overlap}}^{\mathcal{G}}(\beta) \right\}, \tag{1}$$

where $\Psi$ is the $n \times d$ matrix given by the features $\psi_j$ in the dictionary evaluated in the training set points, $[\Psi]_{i,j} = \psi_j(x_i)$. The term $\frac{1}{n} \|\Psi\beta - y\|^2$ is the empirical error, $\frac{1}{n} \sum_{i=1}^n \ell(f(x_i), y_i)$, when the cost function[1] $\ell : \mathbb{R} \times Y \to \mathbb{R}^+$ is the square loss, $\ell(f(x), y) = (y - f(x))^2$.

The penalty term $\Omega_{\text{overlap}}^{\mathcal{G}} : \mathbb{R}^d \to \mathbb{R}_+$ is lower semicontinuous, convex, and one-homogeneous, ($\Omega_{\text{overlap}}^{\mathcal{G}}(\lambda\beta) = \lambda\Omega_{\text{overlap}}^{\mathcal{G}}(\beta), \forall \beta \in \mathbb{R}^d$ and $\lambda \in \mathbb{R}^+$), and is defined as

$$\Omega_{\text{overlap}}^{\mathcal{G}}(\beta) = \inf_{(v_1, \dots, v_B), v_r \in \mathbb{R}^d, \text{supp}(v_r) \subset G_r, \sum_{r=1}^B v_r = \beta} \sum_{r=1}^B \|v_r\|.$$

The functional $\Omega_{\text{overlap}}^{\mathcal{G}}$ was introduced in [12] as a generalization of the group lasso penalty to allow overlapping groups, while maintaining the group lasso property of enforcing sparse solutions which support is a *union of groups*. When groups do not overlap, $\Omega_{\text{overlap}}^{\mathcal{G}}$ reduces to the group lasso

penalty. Note that, as pointed out in [12], using $\sum_{r=1}^{B} \|\beta\|_{G_r}$ as generalization of the group lasso penalty leads to a solution which support is the *complement of the union of groups*. For an extensive study of the properties of $\Omega_{\text{overlap}}^{\mathcal{G}}$, its comparison with the $\ell_1$ norm, and its extension to graph lasso, we therefore refer the interested reader to [12].

## 3 The GLO-pridu Algorithm

If one needs to solve problem (1) for high dimensional data, the use of standard second-order methods such as interior-point methods is precluded (see for instance [6]), since they need to solve large systems of linear equations to compute the Newton steps. On the other hand, first order methods inspired to Nesterov's seminal paper [19] (see also [18]) and based on proximal methods already proved to be a computationally efficient alternative in many machine learning applications [9, 21].

### 3.1 A Proximal algorithm

Given the convex functional $\mathcal{E}_\tau$ in (1), which is sum of a differentiable term, namely $\frac{1}{n} \|\Psi\beta - y\|^2$, and a non-differentiable one-homogeneous term $2\tau\Omega_{\text{overlap}}^{\mathcal{G}}$, its minimum can be computed with following acceleration of the iterative forward-backward splitting scheme

$$\beta^p = \left(I - \pi_{\tau/\sigma K}\right) \left(h^p - \frac{1}{n\sigma} \Psi^T(\Psi h^p - y)\right)$$

$$c_p = (1 - t_p)c_{p-1}, \qquad t_{p+1} = \left(-c_p + \sqrt{c_p^2 + 8c_p}\right)/4 \qquad (2)$$

$$h^{p+1} = \beta^p (1 - t_{p+1} + \frac{t_{p+1}}{t_p}) + \beta^{p-1}(t_p - 1)\frac{t_{p+1}}{t_p}$$

for a suitable choice of $\sigma$. Due to one-homogeneity of $\Omega_{\text{overlap}}^{\mathcal{G}}$, the proximity operator associated to $\frac{\tau}{\sigma}\Omega_{\text{overlap}}^{\mathcal{G}}$ reduces to the identity minus the projection onto the subdifferential of $\frac{\tau}{\sigma}\Omega_{\text{overlap}}^{\mathcal{G}}$ at the origin, which is a closed and convex set. We will denote such a projection as $\pi_{\tau/\sigma K}$, where $K = \partial\Omega_{\text{overlap}}^{\mathcal{G}}(0)$. The above scheme is inspired to [10], and is equivalent to the algorithm named FISTA [5], which convergence is guaranteed, as recalled in the following theorem

**Theorem 1** *Given $\beta^0 \in \mathbb{R}^d$, and $\sigma = \|\Psi^T\Psi\|/n$, let $h^1 = \beta^0$ and $t^1 = 1, c_0 = 1$, then there exists a constant $C_0$ such that the iterative update* (10) *satisfies*

$$\mathcal{E}_\tau(\beta^p) - \mathcal{E}_\tau(\beta^*) \leq \frac{C_0}{p^2}. \qquad (3)$$

As it happens for other accelerations of the basic forward-backward splitting algorithm such as [19, 6, 4], convergence of the sequence $\beta^p$ is no longer guaranteed unless strong convexity is assumed. However, sacrificing theoretical convergence for speed may be mandatory in large scale applications. Furthermore, there is a strong empirical evidence that $\beta^p$ is indeed convergent (see Section 4).

### 3.2 The projection

Note that the proximity operator of the penalty $\Omega_{\text{overlap}}^{\mathcal{G}}$ does not admit a closed form and must be computed approximatively. In fact the projection on the convex set

$$K = \partial\Omega_{\text{overlap}}^{\mathcal{G}}(0) = \{v \in \mathbb{R}^d, \|v\|_{G_r} \leq 1 \text{ for } r = 1, \ldots, B\}.$$

cannot be decomposed group-wise, as in standard group $\ell_1$ regularization, which proximity operator resolves to a group-wise soft-thresholding operator (see Eq. (9) later). Nonetheless, the following lemma shows that, when evaluating the projection, $\pi_K$, we can restrict ourselves to a subset of $\hat{B} = |\hat{\mathcal{G}}| \leq B$ *active* groups. This equivalence is crucial for speeding up the algorithm, in fact $\hat{B}$ is the number of selected groups which is small if one is interested in sparse solutions.

**Lemma 1** *Given $\beta \in \mathbb{R}^d$, $\mathcal{G} = \{G_r\}_{r=1}^B$ with $G_r \subset \{1, \ldots, d\}$, and $\tau > 0$, the projection onto the convex set $\tau K$ with $K = \{v \in \mathbb{R}^d, \|v\|_{G_r} \leq 1 \text{ for } r = 1, \ldots, B\}$ is given by*

$$\begin{aligned} &\textit{Minimize} \quad \|v - \beta\|^2 \\ &\textit{subject to} \quad v \in \mathbb{R}^d, \|v\|_G \leq \tau \text{ for } G \in \hat{\mathcal{G}}. \end{aligned} \qquad (4)$$

*where $\hat{\mathcal{G}} := \{G \in \mathcal{G}, \|\beta\|_G > \tau\}$.*

The proof (given in the supplementary material) is based on the fact that the convex set $\tau K$ is the intersection of cylinders that are all centered on a coordinate subspace. Since $\hat{B}$ is typically much smaller than $d$, it is convenient to solve the dual problem associated to (4).

**Theorem 2** *Given $\beta \in \mathbb{R}^d$, $\{G_r\}_{r=1}^B$ with $G_r \subset \{1, \ldots, d\}$, and $\tau > 0$, the projection onto the convex set $\tau K$ with $K = \{v \in \mathbb{R}^d, \|v\|_{G_r} \leq \tau \text{ for } r = 1, \ldots, B\}$ is given by*

$$[\pi_{\tau K}(\beta)]_j = \frac{\beta_j}{(1 + \sum_{r=1}^{\hat{B}} \lambda_r^* \mathbf{1}_{r,j})} \qquad \text{for } j = 1, \ldots, d \qquad (5)$$

*where $\lambda^*$ is the solution of*

$$\operatorname*{argmax}_{\lambda \in \mathbb{R}_+^{\hat{B}}} f(\lambda), \qquad \text{with } f(\lambda) := \sum_{j=1}^{d} \frac{-\beta_j^2}{1 + \sum_{r=1}^{\hat{B}} \mathbf{1}_{r,j} \lambda_r} - \sum_{r=1}^{\hat{B}} \lambda_r \tau^2, \qquad (6)$$

$\hat{\mathcal{G}} = \{G \in \mathcal{G}, \|\beta\|_G > \tau\} := \{\hat{G}_1, \ldots, \hat{G}_{\hat{B}}\}$, *and $\mathbf{1}_{r,j}$ is $1$ if $j$ belongs to group $\hat{G}_r$ and $0$ otherwise.*

Equation (6) is the dual problem associated to (4), and, since strong duality holds, the minimum of (4) is equal to the maximum of the dual problem, which can be efficiently solved via Bertsekas' projected Newton method described in [7], and here reported as Algorithm 1.

---

**Algorithm 1** Projection

---

**Given:** $\beta \in \mathbb{R}^d, \lambda^{\text{init}} \in \mathbb{R}^{\hat{B}}, \eta \in (0,1), \delta \in (0, 1/2), \epsilon > 0$
**Initialize:** $q = 0, \lambda^0 = \lambda^{\text{init}}$
**while** $(\partial_r f(\lambda^q) > 0$ if $\lambda_r^q = 0$, or $|\partial_r f(\lambda^q)| > \epsilon$ if $\lambda_r^q > 0$, for $r = 1, \ldots, \hat{B})$ **do**
  $q := q + 1$

$$\epsilon_q = \min\{\epsilon, \|\lambda^q - [\lambda^q - \nabla f(\lambda^q)]_+\|\}$$

$$\mathcal{I}_+^q = \{r \text{ such that } 0 \leq \lambda_r^q \leq \epsilon_q, \ \partial_r f(\lambda^q) > 0\}$$

$$H_{r,s} = \begin{cases} 0 & \text{if } r \neq s, \text{ and } r \in \mathcal{I}_+^q \text{ or } s \in \mathcal{I}_+^q \\ \partial_r \partial_s f(\lambda^q) & \text{otherwise} \end{cases} \qquad (7)$$

$$\lambda(\alpha) = [\lambda^q - \alpha (H^q)^{-1} \nabla f(\lambda^q)]_+$$

  $m = 0$
  **while** $f(\lambda^q) - f(\lambda(\eta^m)) \geq \delta \left\{ \eta^m \sum_{r \notin \mathcal{I}_+^q} \partial_r f(\lambda^q) + \sum_{r \in \mathcal{I}_+^q} \partial_r f(\lambda^q) [\lambda_r^q - \lambda_r(\eta^m)] \right\}$ **do**
    $m := m + 1$
  **end while**

$$\lambda^{q+1} = \lambda(\eta^m)$$

**end while**
**return** $\lambda^{q+1}$

---

Bertsekas' iterative scheme combines the basic simplicity of the steepest descent iteration [22] with the quadratic convergence of the projected Newton's method [8]. It does not involve the solution of a quadratic program thereby avoiding the associated computational overhead.

### 3.3 Computing the regularization path

In Algorithm 2 we report the complete **G**roup **L**asso with **O**verlap **pri**mal-**du**al (GLO-pridu) scheme for computing the regularization path, i.e. the set of solutions corresponding to different values of the regularization parameter $\tau_1 > \ldots > \tau_T$, for problem (1). Note that we employ the *continuation* strategy proposed in [11]. A similar warm starting is applied to the inner iteration, where at the $p$-th step $\lambda^{\text{init}}$ is determined by the solution of the $(p-1)$-th projection. Such an initialization empirically proved to guarantee convergence, despite the local nature of Bertsekas' scheme.

### 3.4 The replicates formulation

An alternative way to solve the optimization problem (1) is proposed by [12], where the authors show that problem (1) is equivalent to the standard group $\ell_1$ regularization (without overlap) in an expanded space built by replicating variables belonging to more than one group:

---

**Algorithm 2** GLO-pridu regularization path

---

**Given:** $\tau_1 > \tau_2 > \cdots > \tau_T, \mathcal{G}, \eta \in (0,1), \delta \in (0,1/2), \epsilon_0 > 0, \nu > 0$
**Let:** $\sigma = ||\Psi^T\Psi||/n$
**Initialize:** $\beta(\tau_0) = 0$
**for** $t = 1, \ldots, T$ **do**
    **Initialize:** $\beta^0 = \beta(\tau_{t-1}), \lambda_0^* = 0$
    **while** $||\beta^p - \beta^{p-1}|| > \nu ||\beta^{p-1}||$ **do**
       • $w = h^p - (n\sigma)^{-1}\Psi^T(\Psi h^p - y)$
       • Find $\hat{\mathcal{G}} = \{G \in \mathcal{G}, ||w||_G \geq \tau\}$
       • Compute $\lambda_p^*$ via Algorithm 1 with groups $\hat{\mathcal{G}}$, initialization $\lambda_{p-1}^*$ and tolerance $\epsilon_0 p^{-3/2}$
       • Compute $\beta^p$ as $\beta_j^p = w_j(1 + \sum_{r=1}^{\hat{B}}\lambda_r^{q+1}\mathbf{1}_{r,j})^{-1}$ for $j = 1, \ldots, d$, see Equation (5)
       • Update $c^p, t^p$, and $h^p$ as in (10)
    **end while**
    $\beta(\tau_t) = \beta^p$
**end for**
**return** $\beta(\tau_1), \ldots, \beta(\tau_T)$

---

$$\tilde{\beta}^* \in \operatorname*{argmin}_{\tilde{\beta} \in \mathbb{R}^{\tilde{d}}}\left\{\frac{1}{n}||\tilde{\Psi}\tilde{\beta} - y||^2 + 2\tau\sum_{r=1}^{B}||\tilde{\beta}||_{\tilde{G}_r}\right\}, \tag{8}$$

where $\tilde{\Psi}$ is the matrix built by concatenating copies of $\Psi$ restricted each to a certain group, i.e. $(\tilde{\Psi}_j)_{j \in \tilde{G}_r} = (\Psi_j)_{j \in G_r}$, where $\{\tilde{G}_1, \ldots, \tilde{G}_B\} = \{[1, \ldots, |G_1|], [1+|G_1|, \ldots, |G_1|+|G_2|], \ldots, [\tilde{d}-|G_B|, \ldots, \tilde{d}|]\}$, and $\tilde{d} = \sum_{r=1}^{B}|G_r|$ is the number of total variables obtained after including the replicates. One can then reconstruct $\beta^*$ from $\tilde{\beta}^*$ as $\beta_j^* = \sum_{r=1}^{B}\phi_{G_r}(\tilde{\beta}^*)$, where $\phi_{G_r} : \mathbb{R}^{\tilde{d}} \to \mathbb{R}^d$ maps $\tilde{\beta}$ in $v \in \mathbb{R}^d$, such that $\operatorname{supp}(v) \subset G_r$ and $(v_j)_{j \in G_r} = (\tilde{\beta}_j)_{j \in \tilde{G}_r}$, for $r = 1, \ldots, B$. The main advantage of the above formulation relies on the possibility of using any state-of-the-art optimization procedure for group lasso. In terms of proximal methods, a possible solution is given by Algorithm 3, where $\mathbf{S}_{\tau/\sigma}$ is the proximity operator of the new penalty, and can be computed exactly as

$$\left(\mathbf{S}_{\tau/\sigma}(\tilde{\beta})\right)_j = \left(||\tilde{\beta}||_{\tilde{G}_r} - \frac{\tau}{\sigma}\right)_+\tilde{\beta}_j, \qquad \text{for } j \in \tilde{G}_r, \qquad \text{for } r = 1, \ldots, B. \tag{9}$$

---

**Algorithm 3** GL-prox

---

**Given:** $\tilde{\beta}^0 \in \mathbb{R}^{\tilde{d}}, \tau > 0, \sigma = ||\tilde{\Psi}^T\tilde{\Psi}||/n$
**Initialize:** $p = 0, \tilde{h}^1 = \tilde{\beta}^0, t^1 = 1$
**while** `convergence not reached` **do**
    $p := p + 1$
$$\tilde{\beta}^p = \mathbf{S}_{\tau/\sigma}\left(\tilde{h}^p - (n\sigma)^{-1}\tilde{\Psi}^T(\tilde{\Psi}\tilde{h}^p - y)\right) \tag{10}$$

$$c_p = (1 - t_p)c_{p-1}, \qquad t_{p+1} = \frac{1}{4}(-c_p + \sqrt{c_p^2 + 8c_p})$$

$$\tilde{h}^{p+1} = \tilde{\beta}^p(1 - t_{p+1} + \frac{t_{p+1}}{t_p}) + \tilde{\beta}_{p-1}(t_p - 1)\frac{t_{p+1}}{t_p}$$

**end while**
**return** $\tilde{\beta}^p$

---

Note that in principle, by applying Lemma 1, the group-soft-thresholding operator in (9) can be computed only on the active groups. In practice this does not yield any advantage, since the identification of the active groups has the same computational cost of the thresholding itself.

## 3.5 Computational issues

For both GL-prox and GLO-pridu, the complexity of one iteration is the sum of the complexity of computing the gradient of the data term and the complexity of computing the proximity operator of the penalty term. The former has complexity $O(dn)$ and $O(\tilde{d}n)$ for GLO-pridu and GL-prox,

respectively, for the case $n < d$. One should then add at each iteration, the cost of performing the projection onto $K$. This can be neglected for the case of replicated variables. On the other hand, the time complexity of one iteration for Algorithm 1 is driven by the number of active groups $\hat{B}$. This number is typically small when looking for sparse solutions. The complexity is thus given by the sum of the complexity of evaluating the inverse of the $\hat{B} \times \hat{B}$ matrix $H$, $O(\hat{B}^3)$, and the complexity of performing the product $H^{-1}\nabla g(\lambda)$, $O(\hat{B}^2)$. The worst case complexity would then be $O(\hat{B}^3)$. Nevertheless, in practice the complexity is much lower because matrix $H$ is highly sparse. In fact, Equation (7) tells us that the part of matrix $H$ corresponding to the active set $\mathcal{I}_+$ is diagonal. As a consequence, if $\hat{B} = \hat{B}_- + \hat{B}_+$, where $\hat{B}_-$ is the number of non active constraints, and $\hat{B}_+$ is the number of active constraints, then the complexity of inverting matrix $H$ is at most $O(\hat{B}_+) + O(\hat{B}_-^3)$. Furthermore the $\hat{B}_- \times \hat{B}_-$ non diagonal part of matrix $H$ is highly sparse, since $H_{r,s} = 0$ if $\hat{G}_r \cap \tilde{G}_s = \emptyset$ and the complexity of inverting it is in practice much lower than $O(\hat{B}_-^3)$. The worst case complexity for computing the projection onto $K$ is thus $O(q \cdot \hat{B}_+) + O(q \cdot \hat{B}_-^3)$, where $q$ is the number of iterations necessary to reach convergence. Note that even if, in order to guarantee convergence, the tolerance for evaluating convergence of the inner iteration must decrease with the number of external iterations, in practice, thanks to warm starting, we observed that $q$ is rarely greater than 10 in the experiments presented here.

Concerning the number of iterations required to reach convergence for GL-prox in the replicates formulation, we empirically observed that it requires a much higher number of iterations than GLO-pridu (see Table 3). We argue that such behavior is due to the combination of two occurences: 1) the local condition number of matrix $\tilde{\Psi}$ is 0 even if $\Psi$ is locally well conditioned, 2) the decomposition of $\beta^*$ as $\tilde{\beta}^*$ is possibly not unique, which is required in order to have a unique solution for (8). The former is due to the presence of replicated columns in $\tilde{\Psi}$. In fact, since $\mathcal{E}_\tau$ is convex but not necessarily strictly convex – as when $n < d$ –, uniqueness and convergence is not always guaranteed unless some further assumption is imposed. Most convergence results relative to $\ell_1$ regularization link uniqueness of the solution as well as the rate of convergence of the Soft Thresholding Iteration to some measure of local conditioning of the Hessian of the differentiable part of $\mathcal{E}_\tau$ (see for instance Proposition 4.1 in [11], where the Hessian restricted to the set of relevant variables is required to be full rank). In our case the Hessian for GL-prox is simply $\tilde{H} = 1/n\tilde{\Psi}^T\tilde{\Psi}$, so that, if the relevant groups have non null intersection, then $\tilde{H}$ restricted to the set of relevant variables is by no means full rank. Concerning the latter argument, we must say that in many real world problems, such as bioinformatics, one cannot easily verify that the solution indeed has a unique decomposition. In fact, we can think of trivial examples where the replicates formulation has not a unique solution.

## 4 Numerical Experiments

In this section we present numerical experiments aimed at comparing the running time performance of GLO-pridu with state-of-the-art algorithms. To ensure a fair comparison, we first run some preliminary experiments to identify the fastest codes for group $\ell_1$ regularization with no overlap. We refer to [6] for an extensive empirical and theoretical comparison of different optimization procedures for solving $\ell_1$ regularization. Further empirical comparisons can be found in [15].

### 4.1 Comparison of different implementations for standard group lasso

We considered three algorithms which are representative of the optimization techniques used to solve group lasso: interior-point methods, (group) coordinate descent and its variations, and proximal methods. As an instance of the first set of techniques we employed the publicly available Matlab code at `http://www.di.ens.fr/~fbach/grouplasso/index.htm` described in [1]. For coordinate descent methods, we employed the R-package `grlplasso`, which implements block coordinate gradient descent minimization for a set of possible loss functions. In the following we will refer to these two algorithms as "'GL-IP" and "GL-BCGD". Finally we use our Matlab implementation of Algorithm GL-prox as an instance of proximal methods.

We first observe that the solutions of the three algorithms coincide up to an error which depends on each algorithm tolerance. We thus need to tune each tolerance in order to guarantee that all iterative algorithms are stopped when the level of approximation to the true solution is the same.

Table 1: Running time (mean and standard deviation) in seconds for computing the entire regularization path of GL-IP, GL-BCGD, and GL-prox for different values of $B$, and $n$. For $B = 1000$, GL-IP could not be computed due to memory reasons.

| $n = 100$ | | $B = 10$ | $B = 100$ | $B = 1000$ |
|---|---|---|---|---|
| | GL-IP | $5.6 \pm 0.6$ | $60 \pm 90$ | – |
| | GL-BCGD | $2.1 \pm 0.6$ | $2.8 \pm 0.6$ | $14.4 \pm 1.5$ |
| | GL-prox | $0.21 \pm 0.04$ | $2.9 \pm 0.4$ | $183 \pm 19$ |

| $n = 500$ | | $B = 10$ | $B = 100$ | $B = 1000$ |
|---|---|---|---|---|
| | GL-IP | $2.30 \pm 0.27$ | $370 \pm 30$ | – |
| | GL-BCGD | $2.15 \pm 0.16$ | $4.7 \pm 0.5$ | $16.5 \pm 1.2$ |
| | GL-prox | $0.1514 \pm 0.0025$ | $2.54 \pm 0.16$ | $109 \pm 6$ |

| $n = 1000$ | | $B = 10$ | $B = 100$ | $B = 1000$ |
|---|---|---|---|---|
| | GL-IP | $1.92 \pm 0.25$ | $328 \pm 22$ | – |
| | GL-BCGD | $2.06 \pm 0.26$ | $18 \pm 3$ | $20.6 \pm 2.2$ |
| | GL-prox | $0.182 \pm 0.006$ | $4.7 \pm 0.5$ | $112 \pm 6$ |

Toward this end, we run Algorithm GL-prox with machine precision, $\nu = 10^{-16}$, in order to have a good approximation of the asymptotic solution. We observe that for many values of $n$ and $d$, and over a large range of values of $\tau$, the approximation of GL-prox when $\nu = 10^{-6}$ is of the same order of the approximation of GL-IP with `optparam.tol`$=10^{-9}$, and of GL-BCGD with `tol`$= 10^{-12}$. Note also that with these tolerances the three solutions coincide also in terms of selection, i.e. their supports are identical for each value of $\tau$. Therefore the following results correspond to `optparam.tol` $= 10^{-9}$ for GL-IP, `tol` $= 10^{-12}$ for GL-BCGD, and $\nu = 10^{-6}$ for GL-prox. For the other parameters of GL-IP we used the values used in the demos supplied with the code. Concerning the data generation protocol, the input variables $x = (x_1, \ldots, x_d)$ are uniformly drawn from $[-1, 1]^d$. The labels $y$ are computed using a noise-corrupted linear regression function, i.e. $y = \beta \cdot x + w$, where $\beta$ depends on the first 30 variables, $\beta_j = 1$ if $j = 1, \ldots, 30$, and 0 otherwise, $w$ is an additive gaussian white noise, and the signal to noise ratio is 5:1. In this case the dictionary coincides with the variables, $\Psi_j(x) = x_j$ for $j = 1, \ldots, d$. We then evaluate the entire regularization path for the three algorithms with $B$ sequential groups of 10 variables, ($G_1 = [1, \ldots, 10]$, $G_2 = [11, \ldots, 20]$, and so on), for different values of $n$ and $B$. In order to make sure that we are working on the correct range of values for the parameter $\tau$, we first evaluate the set of solutions of GL-prox corresponding to a large range of 500 values for $\tau$, with $\nu = 10^{-4}$. We then determine the smallest value of $\tau$ which corresponds to selecting less than $n$ variables, $\tau_{min}$, and the smallest one returning the null solution, $\tau_{max}$. Finally we build the geometric series of 50 values between $\tau_{min}$ and $\tau_{max}$, and use it to evaluate the regularization path on the three algorithms. In order to obtain robust estimates of the running times, we repeat 20 times for each pair $n, B$.

In Table 1 we report the computational times required to evaluate the entire regularization path for the three algorithms. Algorithms GL-BCGD and GL-prox are always faster than GL-IP which, due to memory reasons, cannot by applied to problems with more than 5000 variables, since it requires to store the $d \times d$ matrix $\Psi^T \times \Psi$. It must be said that the code for GP-IL was made available mainly in order to allow reproducibility of the results presented in [1], and is not optimized in terms of time and memory occupation. However it is well known that standard second-order methods are typically precluded on large data sets, since they need to solve large systems of linear equations to compute the Newton steps. GL-BCGD is the fastest for $B = 1000$, whereas GL-prox is the fastest for $B = 10, 100$. The candidates as benchmark algorithms for comparison with GLO-pridu are GL-prox and GL-BCGD. Nevertheless we observed that, when the input data matrix contains a significant fraction of replicated columns, this algorithm does not provide sparse solutions. We therefore compare GLO-pridu with GL-prox only.

### 4.1.1 Projection vs duplication

The data generation protocol is equal to the one described in the previous experiments, but $\beta$ depends on the first $12/5b$ variables (which correspond to the first three groups)

$$\beta = (\underbrace{c, \ldots, c}_{b \cdot 12/5 \text{ times}}, \underbrace{0, 0, \ldots, 0}_{d - b \cdot 12/5 \text{ times}}).$$

We then define $B$ groups of size $b$, so that $\tilde{d} = B \cdot b > d$. The first three groups correspond to the subset of relevant variables, and are defined as $G_1 = [1, \ldots, b]$, $G_2 = [4/5b + 1, \ldots, 9/5b]$, and $G_3 = [1, \ldots, b/5, 8/5b + 1, \ldots, 12/5b]$, so that they have a 20% pair-wise overlap. The remaining $B - 3$ groups are built by randomly drawing sets of $b$ indexes from $[1, d]$. In the following we will let $n = 10|G_1 \cup G_2 \cup G_3|$, i.e. $n$ is ten times the number of relevant variables, and vary $d, b$. We also vary the number of groups $B$, so that the dimension of the expanded space is $\alpha$ times the input dimension, $\tilde{d} = \alpha d$, with $\alpha = 1.2, 2, 5$. Clearly this amounts to taking $B = \alpha \cdot d/b$. The parameter $\alpha$ can be thought of as the average number of groups a single variable belongs to. We identify the correct range of values for $\tau$ as in the previous experiments, using GLO-pridu with loose tolerance, and then evaluate the running time and the number of iterations necessary to compute the entire regularization path for GL-prox on the expanded space and GLO-pridu, both with $\nu = 10^{-6}$. Finally we repeat 20 times for each combination of the three parameters $d, b$, and $\alpha$.

Table 2: Running time (mean $\pm$ standard deviation) in seconds for $b = 10$ (top), and $b = 100$ (below). For each $d$ and $\alpha$, the left and right side correspond to GLO-pridu, and GL-prox, respectively.

|  | $\alpha = 1.2$ | | $\alpha = 2$ | | $\alpha = 5$ | |
|---|---|---|---|---|---|---|
| $d=1000$ | $0.15 \pm 0.04$ | $0.20 \pm 0.09$ | $1.6 \pm 0.9$ | $5.1 \pm 2.0$ | $12.4 \pm 1.3$ | $68 \pm 8$ |
| $d=5000$ | $1.1 \pm 0.4$ | $1.0 \pm 0.6$ | $1.55 \pm 0.29$ | $2.4 \pm 0.7$ | $103 \pm 12$ | $790 \pm 57$ |
| $d=10000$ | $2.1 \pm 0.7$ | $2.1 \pm 1.4$ | $3.0 \pm 0.6$ | $4.5 \pm 1.4$ | $460 \pm 110$ | $2900 \pm 400$ |

|  | $\alpha = 1.2$ | | $\alpha = 2$ | | $\alpha = 5$ | |
|---|---|---|---|---|---|---|
| $d=1000$ | $11.7 \pm 0.4$ | $24.1 \pm 2.5$ | $11.6 \pm 0.4$ | $42 \pm 4$ | $13.5 \pm 0.7$ | $1467 \pm 13$ |
| $d=5000$ | $31 \pm 13$ | $38 \pm 15$ | $90 \pm 5$ | $335 \pm 21$ | $85 \pm 3$ | $1110 \pm 80$ |
| $d=10000$ | $16.6 \pm 2.1$ | $13 \pm 3$ | $90 \pm 30$ | $270 \pm 120$ | $296 \pm 16$ | – |

Table 3: Number of iterations (mean $\pm$ standard deviation) for $b = 10$ (top) and $b = 100$ (below). For each $d$ and $\alpha$, the left and right side correspond to GLO-pridu, and GL-prox, respectively.

|  | $\alpha = 1.2$ | | $\alpha = 2$ | | $\alpha = 5$ | |
|---|---|---|---|---|---|---|
| $d=1000$ | $100 \pm 30$ | $80 \pm 30$ | $1200 \pm 500$ | $1900 \pm 800$ | $2150 \pm 160$ | $11000 \pm 1300$ |
| $d=5000$ | $100 \pm 40$ | $70 \pm 30$ | $148 \pm 25$ | $139 \pm 24$ | $6600 \pm 500$ | $27000 \pm 2000$ |
| $d=10000$ | $100 \pm 30$ | $70 \pm 40$ | $160 \pm 30$ | $137 \pm 26$ | $13300 \pm 1900$ | $49000 \pm 6000$ |

|  | $\alpha = 1.2$ | | $\alpha = 2$ | | $\alpha = 5$ | |
|---|---|---|---|---|---|---|
| $d=1000$ | $913 \pm 12$ | $2160 \pm 210$ | $894 \pm 11$ | $2700 \pm 300$ | $895 \pm 10$ | $4200 \pm 400$ |
| $d=5000$ | $600 \pm 400$ | $600 \pm 300$ | $1860 \pm 110$ | $4590 \pm 290$ | $1320 \pm 30$ | $6800 \pm 500$ |
| $d=10000$ | $81 \pm 11$ | $63 \pm 11$ | $1000 \pm 500$ | $1800 \pm 900$ | $2100 \pm 60$ | – |

Running times and number of iterations are reported in Table 2 and 3, respectively. When the degree of overlap $\alpha$ is low the computational times of GL-prox and GLO-pridu are comparable. As $\alpha$ increases, there is a clear advantage in using GLO-pridu instead of GL-prox. The same behavior occurs for the number of iterations.

## 5    Discussion

We have presented an efficient optimization procedure for computing the solution of group lasso with overlapping groups of variables, which allows dealing with high dimensional problems with large groups overlap. We have empirically shown that our procedure has a great computational advantage with respect to state-of-the-art algorithms for group lasso applied on the expanded space built by replicating variables belonging to more than one group. We also mention that computational performance may improve if our scheme is used as core for the optimization step of active set methods, such as [23]. Finally, as shown in [17], the improved computational performance enables to use group $\ell_1$ regularization with overlap for pathway analysis of high-throughput biomedical data, since it can be applied to the entire data set and using all the information present in online databases, without pre-processing for dimensionality reduction.

## Footnotes

[1] Note our analysis would immediately apply to other loss functions, e.g. the logistic loss.

# References

[1] F. Bach. Consistency of the group lasso and multiple kernel learning. *Journal of Machine Learning Research*, 9:1179–1225, 2008.

[2] F. Bach. High-dimensional non-linear variable selection through hierarchical kernel learning. Technical Report HAL 00413473, INRIA, 2009.

[3] F. R. Bach, G. Lanckriet, and M. I. Jordan. Multiple kernel learning, conic duality, and the smo algorithm. In *ICML*, volume 69 of *ACM International Conference Proceeding Series*, 2004.

[4] A. Beck and Teboulle. M. Fast gradient-based algorithms for constrained total variation image denoising and deblurring problems. *IEEE Transactions on Image Processing*, 18(11):2419–2434, 2009.

[5] A. Beck and M. Teboulle. A fast iterative shrinkage-thresholding algorithm for linear inverse problems. *SIAM J. Imaging Sci.*, 2(1):183–202, 2009.

[6] S. Becker, J. Bobin, and E. Candes. Nesta: A fast and accurate first-order method for sparse recovery, 2009.

[7] D. Bertsekas. Projected newton methods for optimization problems with simple constraints. *SIAM Journal on Control and Optimization*, 20(2):221–246, 1982.

[8] R. Brayton and J. Cullum. An algorithm for minimizing a differentiable function subject to. *J. Opt. Th. Appl.*, 29:521–558, 1979.

[9] J. Duchi and Y. Singer. Efficient online and batch learning using forward backward splitting. *Journal of Machine Learning Research*, 10:28992934, December 2009.

[10] O. Guler. New proximal point algorithm for convex minimization. *SIAM J. on Optimization*, 2(4):649–664, 1992.

[11] E. T. Hale, W. Yin, and Y. Zhang. Fixed-point continuation for l1-minimization: Methodology and convergence. *SIOPT*, 19(3):1107–1130, 2008.

[12] L. Jacob, G. Obozinski, and J.-P. Vert. Group lasso with overlap and graph lasso. In *ICML*, page 55, 2009.

[13] R. Jenatton, J.-Y . Audibert, and F. Bach. Structured variable selection with sparsity-inducing norms. Technical report, INRIA, 2009.

[14] R. Jenatton, J. Mairal, G. Obozinski, and F. Bach. Proximal methods for sparse hierarchical dictionary learning. In *Proceeding of ICML 2010*, 2010.

[15] I. Loris. On the performance of algorithms for the minimization of $l_1$-penalized functionals. *Inverse Problems*, 25(3):035008, 16, 2009.

[16] L. Meier, S. van de Geer, and P. Buhlmann. The group lasso for logistic regression. *J. R. Statist. Soc*, B(70):53–71, 2008.

[17] S. Mosci, S. Villa, Verri A., and L. Rosasco. A fast algorithm for structured gene selection. presented at MLSB 2010, Edinburgh.

[18] Y. Nesterov. A method for unconstrained convex minimization problem with the rate of convergence $o(1/k^2)$. *Doklady AN SSSR*, 269(3):543–547, 1983.

[19] Y. Nesterov. Smooth minimization of non-smooth functions. *Math. Prog. Series A*, 103(1):127–152, 2005.

[20] M. Y. Park and T. Hastie. L1-regularization path algorithm for generalized linear models. *J. R. Statist. Soc. B*, 69:659–677, 2007.

[21] L. Rosasco, M. Mosci, S. Santoro, A. Verri, and S. Villa. Iterative projection methods for structured sparsity regularization. Technical Report MIT-CSAIL-TR-2009-050, MIT, 2009.

[22] J. Rosen. The gradient projection method for nonlinear programming, part i: linear constraints. *J. Soc. Ind. Appl. Math.*, 8:181–217, 1960.

[23] V. Roth and B. Fischer. The group-lasso for generalized linear models: uniqueness of solutions and efficient algorithms. In *Proceedings of 25th ICML*, 2008.

[24] P. Zhao, G. Rocha, and B. Yu. The composite absolute penalties family for grouped and hierarchical variable selection. *Annals of Statistics*, 37(6A):3468–3497, 2009.

